# Hyperbolic Self-Organizing Maps for Semantic Navigation

**Jörg Ontrup**
Neuroinformatics Group
Faculty of Technology
Bielefeld University
D-33501 Bielefeld, Germany
*jontrup@techfak.uni-bielefeld.de*

**Helge Ritter**
Neuroinformatics Group
Faculty of Technology
Bielefeld University
D-33501 Bielefeld, Germany
*helge@techfak.uni-bielefeld.de*

## Abstract

We introduce a new type of Self-Organizing Map (SOM) to navigate in the Semantic Space of large text collections. We propose a "hyperbolic SOM" (HSOM) based on a regular tesselation of the hyperbolic plane, which is a non-euclidean space characterized by constant negative gaussian curvature. The *exponentially* increasing size of a neighborhood around a point in hyperbolic space provides more freedom to map the complex information space arising from language into spatial relations. We describe experiments, showing that the HSOM can successfully be applied to text categorization tasks and yields results comparable to other state-of-the-art methods.

## 1   Introduction

For many tasks of exploraty data analysis the Self-Organizing Maps (SOM), as introduced by Kohonen more than a decade ago, have become a widely used tool [1, 2]. So far, the overwhelming majority of SOM approaches have taken it for granted to use a *flat space* as their data model and, motivated by its convenience for visualization, have favored the (suitably discretized) *euclidean plane* as their chief "canvas" for the generated mappings.

However, even if our thinking is deeply entrenched with euclidean space, an obvious limiting factor is the rather restricted neighborhood that "fits" around a point on a euclidean 2D surface. *Hyperbolic spaces* in contrast offer an interesting loophole. They are characterized by uniform negative curvature, resulting in a geometry such that the size of a neighborhood around a point increases *exponentially* with its radius $r$. This exponential scaling behavior allows to create novel displays of large hierarchical structures that are particular accessible to visual inspection [3, 4].

Consequently, we suggest to use hyperbolic spaces also in conjunction with the SOM. The lattice structure of the resulting *hyperbolic SOMs* (HSOMs) is based on a tesselation of the hyperbolic space (in 2D or 3D) and the lattice neighborhood reflects the hyperbolic distance metric that is responsible for the non-intuitive properties of hyperbolic spaces.

After a brief introduction to the construction of hyperbolic spaces we describe several computer experiments that indicate that the HSOM offers new interesting perspectives in the field of text-mining.

## 2   Hyperbolic Spaces

Hyperbolic and spherical spaces are the only non-euclidean geometries that are homogeneous and have isotropic distance metrics [5, 6]. The geometry of H2 is a standard topic in *Riemannian geometry* (see, e.g. [7]), and the relationships for the area $A$ and the circumference $C$ of a circle of radius $r$ are given by

$$A = 4\pi \sinh^2(r/2), \ C = 2\pi \sinh(r) \ . \tag{1}$$

These formulae exhibit the highly remarkable property that both quantities grow *exponentially* with the radius $r$. It is this property that was observed in [3, 4] to make hyperbolic spaces extremely useful for accommodating hierarchical structures.

To use this potential for the SOM, we must solve two problems: $(i)$ we must find suitable discretization lattices on H2 to which we can "attach" the SOM prototype vectors. $(ii)$ after having constructed the SOM, we must somehow project the (hyperbolic!) lattice into "flat space" in order to be able to inspect the generated maps.

### 2.1   Projections of Hyperbolic Spaces

To construct an isometric (i.e., distance preserving) embedding of the hyperbolic plane into a "flat" space, we may use a *Minkowski space* [8]. In such a space, the squared distance $d^2$ between two points $(x, y, u)$ and $(x', y', u')$ is given by

$$d^2 = (x - x')^2 + (y - y')^2 - (u - u')^2 \tag{2}$$

i.e., it ceases to be positive definite. Still, this is a space with zero curvature and its somewhat peculiar distance measure allows to construct an *isometric* embedding of the hyperbolic plane H2, given by

$$x = \sinh(\rho)\cos(\phi), \ y = \sinh(\rho)\sin(\phi), \ u = \cosh(\rho) \ , \tag{3}$$

where $(\rho, \phi)$ are polar coordinates on the H2. Under this embedding, the hyperbolic plane appears as the surface $M$ swept out by rotating the curve $u^2 = 1 + x^2 + y^2$ about the $u$-axis.

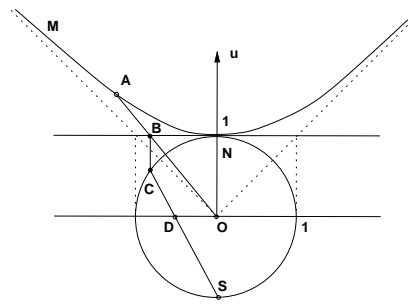

**Figure 1:** Construction steps underlying *Klein* and *Poincaré*-models of the space H2

From this embedding, we can construct two further ones, the so-called *Klein model* and the *Poincaré model* [5, 9] (we will use the latter to visualize HSOMs below). Both achieve a projection of the infinite H2 into the unit disk, however, at the price of distorting distances. The Klein model is obtained by projecting the points of $M$ onto the plane $u = 1$ along rays passing through the origin $O$ (see Fig. 1). Obviously, this projects all points of $M$ into the "flat" unit disk $x^2 + y^2 < 1$ of $\mathbb{R}^2$. (e.g., $A \mapsto B$). The Poincaré Model results if we add two further steps: first a perpendicular projection of the Klein Model onto the ("northern") surface of the unit sphere centered at the origin (e.g., $B \mapsto C$), and then a stereographic projection of the "northern" hemisphere onto the unit circle about the origin in the ground plane $u = 0$ (point $D$). It turns out that the resulting projection of H2 has a number of pleasant properties, among them the preservation of

angles and the mapping of shortest paths onto circular arcs belonging to circles that intersect the unit disk at right angles. Distances in the original H2 are strongly distorted in its Poincaré (and also in the Klein) image (cf. Eq. (5)), however, in a rather useful way: the mapping exhibits a strong "fish-eye"-effect. The neighborhood of the H2 origin is mapped almost faithfully (up to a linear shrinkage factor of 2), while more distant regions become increasingly "squeezed". Since asymptotically the radial distances and the circumference grow both according to the same exponential law, the squeezing is "conformal", i.e., (sufficiently small) shapes painted onto H2 are not deformed, only their size shrinks with increasing distance from the origin. By translating the original H2, the fish-eye-fovea can be moved to any other part of H2, allowing to selectively zoom-in on interesting portions of a map painted on H2 while still keeping a coarser view of its surrounding context.

## 2.2 Tesselations of the Hyperbolic Plane

To complete the set-up for a hyperbolic SOM we still need an equivalent of a regular grid in the hyperbolic plane. For the hyperbolic plane there exist an infinite number of tesselations with congruent polygons such that each grid point is surrounded by the same number $n$ of neighbors [9, 10]. Fig. 2 shows two example tesselations (for the minimal value of $n = 7$ and for $n = 10$), using the Poincaré model for their visualization. While these tesselations appear non-uniform, this is only due to the fish-eye effect of the Poincaré projection. In the original H2, each tesselation triangle has the same size.

One way to generate these tesselations algorithmically is by repeated application of a suitable set of generators of their symmetry group to a (suitably sized, cf. below) "starting triangle", for more details cf. [11].

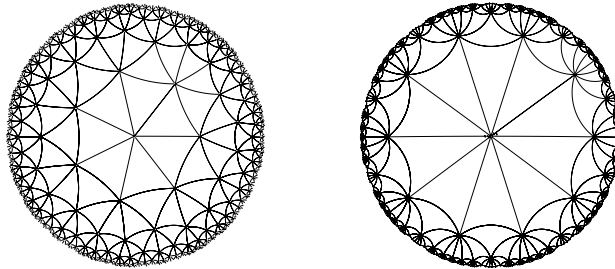

**Figure 2:** Regular triangle tesselations of the hyperbolic plane, projected into the unit disk using the Poincaré mapping. The left tesselation shows the case where the minimal number ($n = 7$) of equilateral triangles meet at each vertex, the right figure was constructed with $n = 10$. In the Poincaré projection, only sides passing through the origin appear straight, all other sides appear as circular arcs, although in the original space all triangles are congruent.

# 3 Hyperbolic SOM Algorithm

We have now all ingredients required for a "hyperbolic SOM". We organize the nodes of a lattice as described above in "rings" around an origin node. The numbers of nodes of such a lattice grows very rapidly (asymptotically exponentially) with the chosen lattice radius $R$ (its number of rings). For instance, a lattice with $n = 7, R = 6$ contains 1625 nodes. Each lattice node $r$ carries a prototype vector $\vec{w}_r \in \mathbb{R}^D$ from some $D$-dimensional feature space (if we wish to make any non-standard assumptions about the metric structure of this space, we would build this into the distance metric that is used for determining the best-match node). The SOM is then formed in the usual way, e.g., in on-line mode by

repeatedly determining the winner node $s$ and adjusting all nodes $r \in N(s,t)$ in a radial lattice neighborhood $N(s,t)$ around $s$ according to the familiar rule

$$\Delta \vec{w}_r = \eta h_{rs}(\vec{x} - \vec{w}_r) \qquad (4)$$

with $h_{rs} = \exp(-d^2(r,s)/2\sigma^2)$. However, since we now work on a hyperbolic lattice, we have to determine both the neighborhood $N(s,t)$ and the (squared) node distance $d^2(r,s)$ according to the natural metric that is inherited by the hyperbolic lattice.

The simplest way to do this is to keep with each node $r$ a complex number $z_r$ to identify its position in the Poincaré model. The node distance is then given (using the Poincaré model, see e.g. [7]) as

$$d = 2\text{arctanh}\left(\left|\frac{z_r - z_s}{1 - \bar{z}_s \cdot z_r}\right|\right) \; . \qquad (5)$$

The neighborhood $N(t,s)$ can be defined as the subset of nodes within a certain graph distance (which is chosen as a small multiple of the neighborhood radius $\sigma$) around $s$.

## 4 Experiments

Some introductory experiments where several examples illustrate the favorable properties of the HSOM as compared to the "standard" euclidean SOM can be found in [11, 12]. A major example of the use of the SOM for text mining is the WEBSOM project [2].

### 4.1 Text Categorization

In order to apply the HSOM to natural text categorization, i.e. the assignment of natural language documents to a number of predefined categories, we follow the widely used vector-space-model of Information Retrieval (IR). For each document $d$ we construct a feature vector $\vec{f}(d)$, where the components $f_i$ are determined by the frequency of which term $t_i$ occurs in that document. Following standard practice [13] we choose a *term frequency* × *inverse document frequency* weighting scheme:

$$f_i = tf(t_i, j) \, log\left(\frac{N}{df(t_i)}\right) \; , \qquad (6)$$

where the term frequency $tf(t_i, j)$ denotes the number of times term $t_i$ occurs in $d_j$, $N$ the number of documents in the training set and $df(t_i)$ the document frequency of $t_i$, i.e. the number of documents $t_i$ occurs in.

The HSOM can be utilized for text categorization in the following manner. In a first step, the training set is used to adapt the weight vectors $\vec{w}_r$ according to (4). During the second step, the training set is mapped onto the HSOM lattice. To this end, for each training example $d_j$ its best match node $s$ is determined such that

$$\left|\vec{f}(d_j) - \vec{w}_s\right| \leqslant \left|\vec{f}(d_j) - \vec{w}_r\right| \quad \forall r \; , \qquad (7)$$

where $\vec{f}(d_j)$ denotes the feature vector of document $d_j$, as described above. After all examples have been presented to the net, each node is labelled with the union $U_r$ of all categories that belonged to the documents that were mapped to this node. A new, unknown text is then classified into the union $U_s$ of categories which are associated with its winner node $s$ selected in the HSOM.

**Text Collection.** We used the Reuters-21578[1] data set since it provides a well known baseline which is also used by other authors to evaluate their approaches, c.f. [14, 15]. We

have used the "ModApte" split, leading to 9603 training and 3299 test documents. After preprocessing, our training set contained 5561 distinct terms.

**Performance Evaluation.** The classification effectiveness is commonly measured in terms of precision $P$ and recall $R$ [16], which can be estimated as $P_i = \frac{TP_i}{TP_i+FP_i}$, $R_i = \frac{TP_i}{TP_i+FN_i}$, where $TP_i$ and $TN_i$ are the numbers of documents correctly classified, and correctly not classified to category $c_i$, respectively. Analogous, $FP_i$ and $FN_i$ are the corresponding numbers of falsely classified documents.

For each node $r$ and each category $c_i$ a confidence value $C_{ri}$ is determined. It describes the number of training documents belonging to class $c_i$ which were mapped to node $r$. When retrieving documents from a given category $c_i$, we compare for each node $r$ its associated $C_{ri}$ against a threshold $\Theta$. Documents from nodes with $C_{ri} > \Theta$ become then included into the retrieval set. For nodes $r$ which contain a set of documents $D(r)$, the order of the retrieval set is ranked by $cos(\vec{f}(d_j), \vec{w}_r)$, where $\vec{f}(d_j) \in D(r)$.

In this way the number of retrieved documents can be controlled and we obtain the precision-recall-diagrams as shown in Fig. 3.

In order to compare the HSOM's performance for text categorization, we also evaluated a $k$-nearest neighbor ($k$-NN) classifier with our training set. Apart from boosting methods [16] only support vector machines [14] have shown better performances. The confidence level of a $k$-NN classifier to assign document $d_j$ to class $c_i$ is

$$C_i^{k\text{-NN}}(d_j) = \sum_{d_z \in N_k(d_j)} a_{iz} \cdot cos(d_j, d_z) , \tag{8}$$

where $N_k(d_j)$ is the set of $k$ documents $d_z$ for which $cos(d_j, d_z)$ is maximum. The assignment factor $a_{iz}$ is 1, if $d_z$ belongs to category $c_i$ and 0 otherwise. According to [14, 17] we have chosen the $k = 30$ nearest neighbors.

**Text Categorization Results.** The results of three experiments are shown in Table 1. We have compared a HSOM with $R = 4$ rings and a tesselation with $n = 9$ neighbors (summing up to 1306 nodes) to a spherical standard euclidean SOM as described in [11] with approx. 1300 nodes, and the $k$-NN classifier. Our results indicate that the HSOM does not perform better than a $k$-NN classifier, but to a certain extent also does not play significantly worse either. It is noticable that for less dominant categories the HSOM yields superior results to those of the standard SOM. This is due to the fact, that the nodes in H2 cover a much broader space and therefore offer more freedom to map smaller portions of the original dataspace with less distortions as compared to euclidean space.

As the $k$-NN results suggest, other state-of-the-art techniques like support vector machines will probably lead to better numerical categorization results than the HSOM. However, since the main purpose of the HSOM is the visualization of relationships between texts and text categories, we believe that the observed categorization performance of the HSOM compares sufficiently well with the more specialized (non-visualization) techniques to warrant its efficient use for creating insightful maps of large bodies of document data.

**Table 1:** Precision-recall breakeven points for the ten most prominent categories.

|  | earn | acq | mny-fx | crude | grain | trade | interest | wheat | ship | corn |
|---|---|---|---|---|---|---|---|---|---|---|
| SOM | 90.0 | 81.2 | 61.7 | 70.3 | 69.4 | 48.8 | 57.1 | 61.9 | 54.8 | 50.3 |
| HSOM | 90.2 | 81.6 | 68.7 | 78.8 | 76.2 | 56.8 | 66.4 | 69.3 | 61.8 | 53.6 |
| $k$-NN | 93.8 | 83.7 | 69.3 | 84.7 | 81.9 | 61.9 | 71.0 | 69.0 | 77.5 | 67.9 |

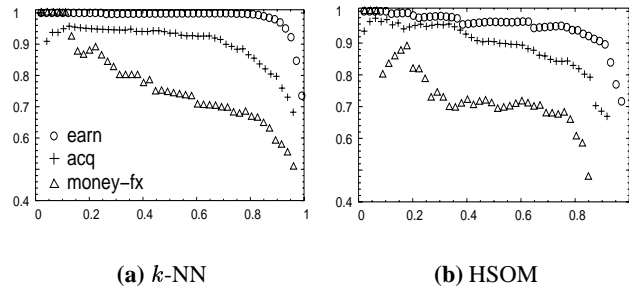

**(a)** $k$-NN          **(b)** HSOM

**Figure 3:** Precision-recall curves for the three most frequent categories *earn*, *acq* and *money-fx*.

## 4.2  Text Mining & Semantic Navigation

A major advantage of the HSOM is its remarkable capability to map high-dimensional similarity relationships to a low-dimensional space which can be more easily handled and interpreted by the human observer. This feature and the particular "fish-eye" capability motivates our approach to visualize whole text collections with the HSOM. It can be regarded as an interface capturing the semantic structure of a text database and provides a way to guide the users attention. In preliminary experiments we have labelled the nodes with glyphs corresponding to the categories of the documents mapped to that node. In Fig. 4 two HSOM views of the Reuters data set are shown. Note, that the major amount of data gets mapped to the outermost region, where the nodes of the HSOM make use of the large space offered by the hyperbolic geometry. During the unsupervised training process, the document's categories were not presented to the HSOM. Nevertheless, several document clusters can be clearly identified. The two most prominent are the *earn* and *acquisition* region of the map, reflecting the large proportion of these categories in the Reuters-21578 collection. Note, that categories which are semantically similar are located beside each other, as can be seen in the *corn*, *wheat*, *grain* the *interest*, *money-fx* or the *crude*, *ship* area of the map. Additional to the category (glyph type) and the number of training documents per node (glyph size), the number of test documents mapped to each node is shown as the height of the symbol above the ground plane. In this way the HSOM can be used as a novelty detector in chronological document streams. For the Reuters-21578 dataset, a particular node strikes out. It corresponds to the small glyph tagged with the "ship" label in Fig. 4. Only a few documents from the training collection are mapped to that node as shown by it's relatively small glyph size. The large $z$-value on the other hand indicates that it contains a large number of test documents, and is therefore probably semantically connected to a significant, novel event only contained in the test collection. The right image of Fig. 4 shows the same map, but the focal view now moved into the direction of the conspicious "ship" node, resulting in a magnification of the corresponding area. A closer inspection reveals, that the vast majority (35 of 40) of the test documents describe an incident where an Iranian oil rig was attacked in the gulf. Although no document of the training set describes this incident (because the text collection is ordered by time and the attack took place "after" the split into train and test set), the HSOM generalizes well and maps the semantic content of these documents to the proper area of the map, located between the regions for *crude* and *ship*.

The next example illustrates that the HSOM can provide more information about an unknown text than just it's category. For this experiment we have taken movie reviews from the rec.art.movies.reviews newsgroup. Since all the reviews describe a certain movie, we retrieved their associated genres from the Internet Movie Database (http://www.imdb.com) to build a set of category labels for each document. The training set contained 8923 ran-

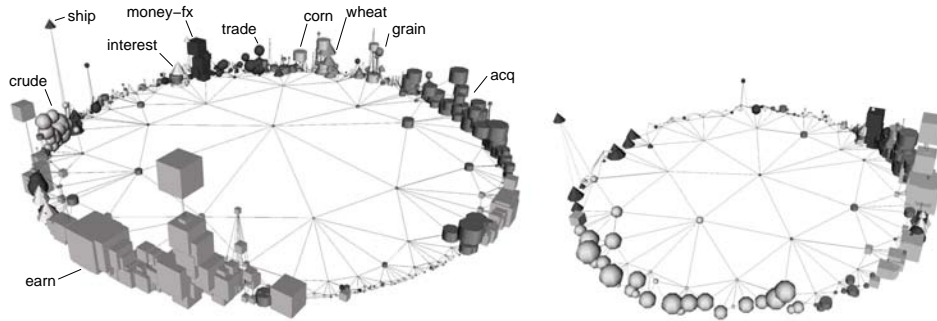

**Figure 4:** The left figure shows a central view of the Reuters data. We used a HSOM with $R = 4$ rings and a tesselation with $n = 9$ neighbors. Ten different glyphs were used to visualize the ten most frequent categories. They were manually tagged to indicate the correspondence between category and symbol type. The glyph sizes and the $z$-values (height above ground plane) reflect the number of training and test documents mapped to the corresponding node, respectively.

domly selected reviews (without their genre information) from films released before 2000. We then presented the system with five reviews from the film "Atlantis", a Disney cartoon released in 2001. The HSOM correctly classified all of the five texts as reviews for an animation movie. In Fig. 5 the projection of the five new documents onto the map with the previously acquired text collection is shown. It can be seen that there exist several clusters related to the animation genre. By moving the fovea of the HSOM we can now "zoom" into that region which contains the five new texts. In the right of Fig. 5 it can be seen that all of the "Atlantis" reviews where mapped to a node in immediate vicinity of documents describing other Disney animation movies. This example motivates the approach of "semantic navigation" to rapidly visualize the linkage between unknown documents and previously acquired semantic concepts.

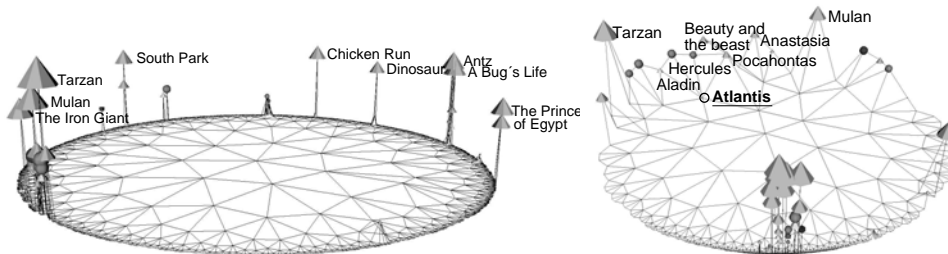

**Figure 5:** A HSOM with $R = 6$ and a tesselation with $n = 7$ neighbors was used to map movie rewies from newsgroup channels. In both figures, glyph size and $z$-value indicate the number of texts related to the animation genre mapped to the corresponding node. Nodes exceeding a certain threshold were labelled with the title corresponding to the most frequently occuring movie mapped to that node. The underlined label in the right figure indicates the position of the node to which five new documents were mapped to.

## 5 Conclusion

Efficient navigation in "Semantic Space" requires to address two challenges: *(i)* how to create a low dimensional display of semantic relationship of documents, and *(ii)* how to obtain these relationships by automated text categorization. Our results show that the HSOM can provide a good solution to both demands simultaneously and within a single framework.

The HSOM is able to exploit the peculiar geometric properties of hyperbolic space to successfully compress complex semantic relationships between text documents. Additionally, the use of hyperbolic lattice topology for the arrangement of the HSOM nodes offers new and attractive features for interactive "semantic navigation". Large document databases can be inspected at a glance while the HSOM provides additional information which was captured during a previous training step, allowing e.g. to rapidly visualize relationships between new documents and previously acquired collections.

Future work will address more sophisticated visualization strategies based on the new approach, as well as the exploration of other text representations which might take advantage of hyperbolic space properties.

## Footnotes

[1]As compiled by David Lewis from the AT&T Research Lab in 1987. The data can be found at http://www.research.att.com/~lewis/

## References

[1] T. Kohonen. *Self-Organizing Maps*. Springer Series in Information Sciences. 3rd edition, 2001.

[2] Teuvo Kohonen, Samuel Kaski, Krista Lagus, Jarkko Salojärvi, Vesa Paatero, and Antti Saarela. Organization of a massive document collection. *IEEE Transactions on Neural Networks, Special Issue on Neural Networks for Data Mining and Knowledge Discovery*, 11(3):574–585, May 2000.

[3] John Lamping and Ramana Rao. Laying out and visualizing large trees using a hyperbolic space. In *Proceedings of UIST'94*, pages 13–14, 1994.

[4] T. Munzer. Exploring large graphs in 3D hyperbolic space. *IEEE Computer Graphics and Applications*, 18(4):18–23, July/August 1998.

[5] H. S. M. Coxeter. *Non Euclidean Geometry*. Univ. of Toronto Press, Toronto, 1957.

[6] J.A. Thorpe. *Elementary Topics in Differential Geometry*. Springer-Verlag, New York, 1979.

[7] Frank Morgan. *Riemannian Geometry: A Beginner's Guide*. Jones and Bartlett Publishers, Boston, London, 1993.

[8] Charles W. Misner, J. A. Wheeler, and Kip S. Thorne. *Gravitation*. Freeman, 1973.

[9] R. Fricke and F. Klein. *Vorlesungen über die Theorie der automorphen Funktionen*, volume 1. Teubner, Leipzig, 1897. Reprinted by Johnson Reprint, New York, 1965.

[10] W. Magnus. *Noneuclidean Tesselations and Their Groups*. Academic Press, 1974.

[11] Helge Ritter. Self-organizing maps in non-euclidian spaces. In E. Oja and S. Kaski, editors, *Kohonen Maps*, pages 97–108. Amer Elsevier, 1999.

[12] J. Ontrup and H. Ritter. Text categorization and semantic browsing with self-organizing maps on non-euclidean spaces. In *Proc. of the PKDD-01*, 2001.

[13] G. Salton and C. Buckley. Term-weighting approaches in automatic text retrieval. *Information Processing and Management*, 24(5):513–523, 1988.

[14] T. Joachims. Text categorization with support vector machines: learning with many relevant features. In *Proc. of ECML-98*, number 1398, pages 137–142, Chemnitz, DE, 1998.

[15] Huma Lodhi, John Shawe-Taylor, Nello Cristianini, and Chris Watkins. Text classification using string kernels. In Todd K. Leen, Thomas G. Dietterich, and Volker Tresp, editors, *Advances in Neural Information Processing Systems 13*, pages 563–569. MIT Press, 2001.

[16] F. Sebastiani, A. Sperduti, and N. Valdambrini. An improved boosting algorithm and its application to automated text categorization. In *Proc. of CIKM-00*, pages 78–85, 2000.

[17] Y. Yang. An evaluation of statistical approaches to text categorization. *Information Retrieval*, 1-2(1):69–90, 1999.
